# An improved estimator of Variance Explained in the presence of noise

**Ralf. M. Haefner**[*]
Laboratory for Sensorimotor Research
National Eye Institute, NIH
Bethesda, MD 20892
ralf.haefner@gmail.com

**Bruce. G. Cumming**
Laboratory for Sensorimotor Research
National Eye Institute, NIH
Bethesda, MD 20892
bgc@lsr.nei.nih.gov

## Abstract

A crucial part of developing mathematical models of information processing in the brain is the quantification of their success. One of the most widely-used metrics yields the percentage of the variance in the data that is explained by the model. Unfortunately, this metric is biased due to the intrinsic variability in the data. We derive a simple analytical modification of the traditional formula that significantly improves its accuracy (as measured by bias) with similar or better precision (as measured by mean-square error) in estimating the true underlying Variance Explained by the model class. Our estimator advances on previous work by a) accounting for overfitting due to free model parameters mitigating the need for a separate validation data set, b) adjusting for the uncertainty in the noise estimate and c) adding a conditioning term. We apply our new estimator to binocular disparity tuning curves of a set of macaque V1 neurons and find that on a population level almost all of the variance unexplained by Gabor functions is attributable to noise.

## 1 Introduction

Constructing models of biological systems, e.g. in systems neuroscience, mostly aims at providing functional descriptions, not fundamental physical laws. It seems likely that any parametric model of signal processing in single neurons can be ruled out given a sufficient amount of data. Rather than only testing the statistical validity of a particular mathematical formulation against data, e.g. by using a $\chi^2$-test, it is equally important to know how much of the signal, or variance, in the data is explained by the model. This is commonly measured by *Variance Explained* (VE), the coefficient of determination or $r^2$ statistic. A fundamental problem of the traditional estimator for VE is its bias in the presence of noise in the data. This noise may be due to measurement error or sampling noise owing to the high intrinsic variability in the underlying data. This is especially important when trying to model cortical neurons where variability is ubiquitous. Either kind of noise is in principle unexplainable by the model and hence needs to be accounted for when evaluating the quality of the model. Since the total variance in the data consists of the *true* underlying variance plus that due to noise, the traditional estimator yields a systematic underestimation of the *true* VE of the model in the absence of noise [1][2][3].

This has been noted by several authors before us; David & Gallant compute the traditional measure at several noise levels and extrapolate it to the noise-free condition [1]. This method relies on many repeats of the same stimulus and is therefore often impractical. Sahani & Linden add an analytical correction to the traditional formula in order to reduce its bias [2]. A number of subsequent studies have used their corrections to evaluate their models (e.g. [4][5][6]). We further improve on Sahani

---

[*]Corresponding author (ralf.haefner@gmail.com)

& Linden's formula in three ways: 1) most importantly by accounting for the number of parameters in the model, 2) adding a correction term for the uncertainty in the noise estimation, and 3) including a conditioning term to improve the performance in the presence of excessive noise. We propose a principled method to choose the conditioning term in order to electively minimize either the bias or the mean-square-error (MSE) of the estimator.

In numerical simulations we find that the analytical correction alone is capable of drastically reducing the bias at moderate and high noise levels while maintaining a mean-square-error about as good as the traditional formula. Only for very high levels of noise is it advantageous to make use of the conditioning term. We test the effect of our improved formula on a data set of disparity selective macaque V1 neurons and find that for many cells noise accounts for most of the unexplained variance. On a population level we find that after adjusting for the noise, Gabor functions can explain about 98% of the underlying response variance.

## 2 Derivation of an improved estimator

### 2.1 Traditional Variance Explained

Given a set of $N$ measurements $d_i$ of process $D$ and given the model predictions $m_i$, the traditional Variance Explained $\nu$ is computed as the difference of total variance $\mathrm{var}(d_i)$ and the variance of the residuals of the model $\mathrm{var}(d_i - m_i)$. It is usually reported as a fraction of total variance:

$$\nu = \frac{\mathrm{var}(d_i) - \mathrm{var}(d_i - m_i)}{\mathrm{var}(d_i)} = 1 - \frac{\mathrm{var}(d_i - m_i)}{\mathrm{var}(d_i)} = 1 - \frac{\sum_{i=1}^{N}(d_i - m_i)^2}{\sum_{i=1}^{N}(d_i - \bar{d})^2}. \tag{1}$$

In most cases, the $d_i$ themselves are averages of individual measurements and subject to a sampling error. Since the variances of independent random variables add, this measurement noise leads to additive noise terms in both numerator and denominator of equation (1). Below we show that as the noise level increases, $\nu \to (n-1)/(N-1)$ with $n$ being the number of model parameters (see equation 8). The consequence is a systematic misestimation of the true Variance Explained (typically underestimation since $(n-1)/(N-1)$ is usually smaller than the true VE). The effect of this can be seen in Figure 1 for two example simulations. In each simulation we fit a model to simulated noisy data sampled from a different but known underlying function. This allows us to compare the estimated VE to the true one, in the absence of noise. The average bias (estimated VE minus true VE) of the traditional variance explained is shown for 2000 instantiations of each simulation (shown in triangles). As we simulate an increase in sampling noise, the variance explained decreases significantly, underestimating the true VE by up to 30% in our examples.

### 2.2 Noise bias

Let $\bar{d}_i = 1/R_i \sum_{j=1}^{R_i} d_{ij}$ where the $R_i$ are the number of observations for each variable $i$. We further assume that the measured $d_{ij}$ are drawn from a Gaussian distribution around the *true* means $D_i$ with a variance of $R\Theta_i^2$. Then the $\bar{d}_i$ are drawn from $\mathcal{N}[D_i; \Theta_i^2]$. To simplify the presentation we assume that the variables have been transformed to equalize all $\Sigma \equiv \Sigma_i$ and that $R \equiv R_i$. It follows that $\sigma^2 = 1/(RN(R-1)) \sum_{i=1}^{N} \sum_{j=1}^{R}(d_{ij} - \bar{d}_i)^2$ is an estimate of $\Theta^2$ based on measurements with $N_\sigma = N(R-1)$ degrees of freedom. In the terms of Sahani & Linden [2], $\sigma^2$ is the noise power. Our estimator, however, is more direct and accurate – especially for small $N$ and $R$.

Let $M_i$ be the best fitting model to $D_i$ of a given model class with parameters. Then the variance explained in the absence of noise becomes:

$$\nu_0 = 1 - \frac{\mathrm{var}(M_i - D_i)}{\mathrm{var}(D_i)} = 1 - \frac{\sum_{i=1}^{N}(D_i - M_i)^2}{\sum_{i=1}^{N}(D_i - \bar{D})^2} \tag{2}$$

where $\bar{D} = 1/N \sum_{i=1}^{N} D_i$. Then $\nu_0$ is the *true* value for the Variance Explained that one would like to know: based on the best fit of the model class to the underlying data in the absence of any measurement or sampling noise. $\nu_0$ is of course unknown and the values obtained by (1) are drawn from a probability distribution around the true Variance Explained.

Normalizing both denominator and numerator of formula (1) by $\sigma^2$ leaves $\nu$ unchanged. However it becomes clear that the resulting denominator is drawn from a noncentral $F$-distribution:

$$\frac{1}{N-1} \sum_{i=1}^{N} \frac{(d_i - \bar{\bar{d}})^2}{\sigma^2} = \frac{\frac{1}{N-1} \sum_{i=1}^{N} (d_i - \bar{\bar{d}})^2 / \Sigma^2}{\frac{1}{N_\sigma} \sum_{i=1}^{N} \sum_{j=1}^{R} (d_{ij} - \bar{d}_i)^2 / (R\Sigma^2)} \sim \frac{\chi_{N-1}^2(\lambda_{\text{DD}})/(N-1)}{\chi_{N_\sigma}^2/N_\sigma} \tag{3}$$

with $N-1$ and $N_\sigma = N(R-1)$ degrees of freedom, the noncentrality parameter $\lambda_{\text{DD}} = \sum_{i=1}^{N} (D_i - \bar{D})^2 / \Sigma^2$ and $\bar{\bar{d}} = 1/N \sum_{i=1}^{N} \bar{d}_i$. For $N_\sigma > 2$ the mean of this distribution is given by

$$\mathrm{E}\left[ \frac{1}{N-1} \sum_{i=1}^{N} \frac{(d_i - \bar{\bar{d}})^2}{\sigma^2} \right] = \frac{N_\sigma(N-1+\lambda_{\text{DD}})}{(N-1)(N_\sigma-2)} \tag{4}$$

Hence, an unbiased estimator of $\sum_{i=1}^{N} (D_i - \bar{D})^2 / \Sigma^2 = \lambda_{\text{DD}}$ is given by

$$\lambda_{\text{DD}} = \frac{N_\sigma - 2}{N_\sigma} \sum_{i=1}^{N} \frac{(d_i - \bar{\bar{d}})^2}{\sigma^2} - (N-1) \tag{5}$$

With the same reasoning we find that the numerator of equation (1)

$$\frac{1}{N-n} \sum_{i=1}^{N} \frac{(d_i - m_i)^2}{\sigma^2} \sim \frac{\chi_{N-n}^2(\lambda_{\text{DD}})/(N-n)}{\chi_{N_\sigma}^2/N_\sigma} \tag{6}$$

follows a noncentral $F$-distribution with $N - n$ and $N_\sigma$ degrees of freedom and the noncentrality parameter $\lambda_{\text{DM}} = \sum_{i=1}^{N} (D_i - M_i)^2 / \Sigma^2$. Hence, an unbiased estimator of $\sum_{i=1}^{N} (D_i - M_i)^2 / \Sigma^2 = \lambda_{\text{DM}}$ is given by

$$\lambda_{\text{DM}} = \frac{N_\sigma - 2}{N_\sigma} \sum_{i=1}^{N} \frac{(d_i - m_i)^2}{\sigma^2} - (N-n) \tag{7}$$

Combining (5) and (7) yields an estimator for $\nu_0$ whose numerator and denominator are individually unbiased:

$$\Upsilon[\nu_0] = 1 - \frac{\sum_{i=1}^{N} \left( \frac{d_i - m_i}{\sigma} \right)^2 - \frac{N_\sigma(N-n)}{N_\sigma - 2}}{\sum_{i=1}^{N} \left( \frac{d_i - \bar{d}}{\sigma} \right)^2 - \frac{N_\sigma(N-1)}{N_\sigma - 2}}. \tag{8}$$

Note that apart from the difference in noise estimation, the estimator proposed by Sahani & Linden is contained in ours as a special case, becoming identical when there is no uncertainty in the noise estimate ($N_\sigma \to \infty$) and testing a model with no free parameters ($n = 0$). $N_\sigma \to \infty$ is an excellent approximation in their case of fitting receptive fields to long series of data, but less so in the case of fitting tuning curves with a limited number of data points. However, the fact that their noise-term does not account for overfitting due to free parameters in the model means that their formula overestimates the true Variance Explained. Hence, it requires a separate validation data set which might be costly to obtain.

At this point we wish to note that (5), (7) and (8) readily generalize to cases where the noise level $\Sigma_i$ and the number of observations $R_i$ on which the means $\bar{d}_i$ are based (and therefore $N_{\sigma_i}$) differ between those data points.

## 2.3 Conditioning term

First it is important to note that while both numerator and denominator in formula (8) are now unbiased, the ratio is generally not. In fact, the ratio is not even well-defined for arbitrary measurements since the denominator can become zero and negative. In practice this is avoided by implicit or explicit selection criteria imposed by the experimenter requiring a minimum SNR in the data before further analysis. An example would be a criterion based on the significance level $p_{\mathrm{ANOVA}}$ of the modulation in the data as assessed by a 1-way ANOVA test. (Any criterion can be used in the context of the framework described here, as long as it is used consistently.) The effect of such a criterion is to cut off the lower tail of the distribution from which the denominator is drawn to exclude zero. This introduces a bias to the denominator the size of which depends on the amount of noise and the strictness of the criterion used. We recognize that both biases are strongest when the data is such that the ratio is close to singular and therefore propose an additive conditioning term $C$ in the denominator of (8):

$$\Upsilon(C) = 1 - \left[ \sum_{i=1}^{N} \left( \frac{d_i - m_i}{\sigma} \right)^2 - \frac{N_\sigma(N-n)}{N_\sigma - 2} \right] \Big/ \left[ \sum_{i=1}^{N} \left( \frac{d_i - \bar{d}}{\sigma} \right)^2 - \frac{N_\sigma(N-1)}{N_\sigma - 2} + C \right]. \quad (9)$$

Depending on the application, the optimal $C$ can be chosen to either minimize the mean-square-error (MSE) $E[\Upsilon(C) - \nu_0]$ or the bias $|E[\Upsilon(C)] - \nu_0|$ of the estimator. Generally, the optimal levels of conditioning for the two scenarios are different, i.e. unbiasedness comes at the expense of an increased MSE and vice versa. For individual estimates a small bias can be acceptable in order to improve accuracy (and hence minimize MSE). When averaging over a large number of estimates, e.g. from a population of neurons, it becomes important that the estimator is unbiased.

$C = C(N, n, N_\sigma, \lambda_{\mathrm{DM}}, \lambda_{\mathrm{DD}}; p_{\mathrm{ANOVA}})$ is itself a function of a number of variables, only two of which, $\lambda_{\mathrm{DM}}$ and $\lambda_{\mathrm{DD}}$, are unknown a priori. We approximate them by our estimates from equations (5) and (7). The optimal $C$ can then be determined in each case by a simple minimization across a large number of random samples drawn from the appropriate distributions (compare equations (3) and (6)):

$$C_{\mathrm{bias}} \quad : \quad \min_{C} \left| \mathrm{E}\left[\Upsilon(C)\right] - (1 - \lambda_{\mathrm{DM}}/\lambda_{\mathrm{DD}}) \right| \quad \text{and therefore:} \quad (10)$$

$$C_{\mathrm{bias}} \quad : \quad \min_{C} \left| \mathrm{E}\left[ \frac{\chi^2_{N-n}(\lambda_{\mathrm{DM}})/\chi^2_{N_\sigma} - (N-n)/(N_\sigma - 2)}{\chi^2_{N-1}(\lambda_{\mathrm{DD}})/\chi^2_{N_\sigma} - (N-1)/(N_\sigma - 2) + C/N_\sigma} \right] - \frac{\lambda_{\mathrm{DM}}}{\lambda_{\mathrm{DD}}} \right| \quad (11)$$

$$C_{\mathrm{MSE}} \quad : \quad \min_{C} \mathrm{E}\left[ \left( \frac{\chi^2_{N-n}(\lambda_{\mathrm{DM}})/\chi^2_{N_\sigma} - (N-n)/(N_\sigma - 2)}{\chi^2_{N-1}(\lambda_{\mathrm{DD}})/\chi^2_{N_\sigma} - (N-1)/(N_\sigma - 2) + C/N_\sigma} - \frac{\lambda_{\mathrm{DM}}}{\lambda_{\mathrm{DD}}} \right)^2 \right] \quad (12)$$

Note that the $\chi^2_{N_\sigma}$ distributions in numerator and denominator, sampling over varying estimates of the underlying noise $\sigma^2$, are shared in both formulas since the $\sigma^2$ is shared. Those two minimization problems can easily be solved by Monte-Carlo sampling the probability distributions and subsequently find the minimum of MSE or bias, respectively, across all samples.

## 2.4 Application to simulated data

Figure 1 demonstrates the performance of various estimators of VE for three synthetic examples. In the left column we show the results when testing a model that consists of a 3rd degree polynomial that has been fit to noisy data sampled from a Gaussian distribution around an underlying sine-function. Over the domain studied here, the true VE of the model as fit to the data in the noiseless condition would be 77%. The center & right column shows the case of a Gabor function that is fit to noisy data sampled from a difference-of-Gaussians "reality". Here the true VE is 90%. The center column simulates Gaussian and the right column Gamma noise (Fano factor of 2).

We confirm that the traditional VE measure (triangles) has an increasingly negative bias with increasing noise level $\sigma$. Applying the Sahani-Linden correction (squares) this negative bias is turned into a positive one since the overfitting of noise due to the free parameters in the model is not taken into consideration. This leads to an overestimation of the true VE when applied to the fitting data instead of a separate set of validation data. Accounting for the number of parameters greatly reduces the bias to close to zero across a large range of noise levels (dots). The bias becomes notable only

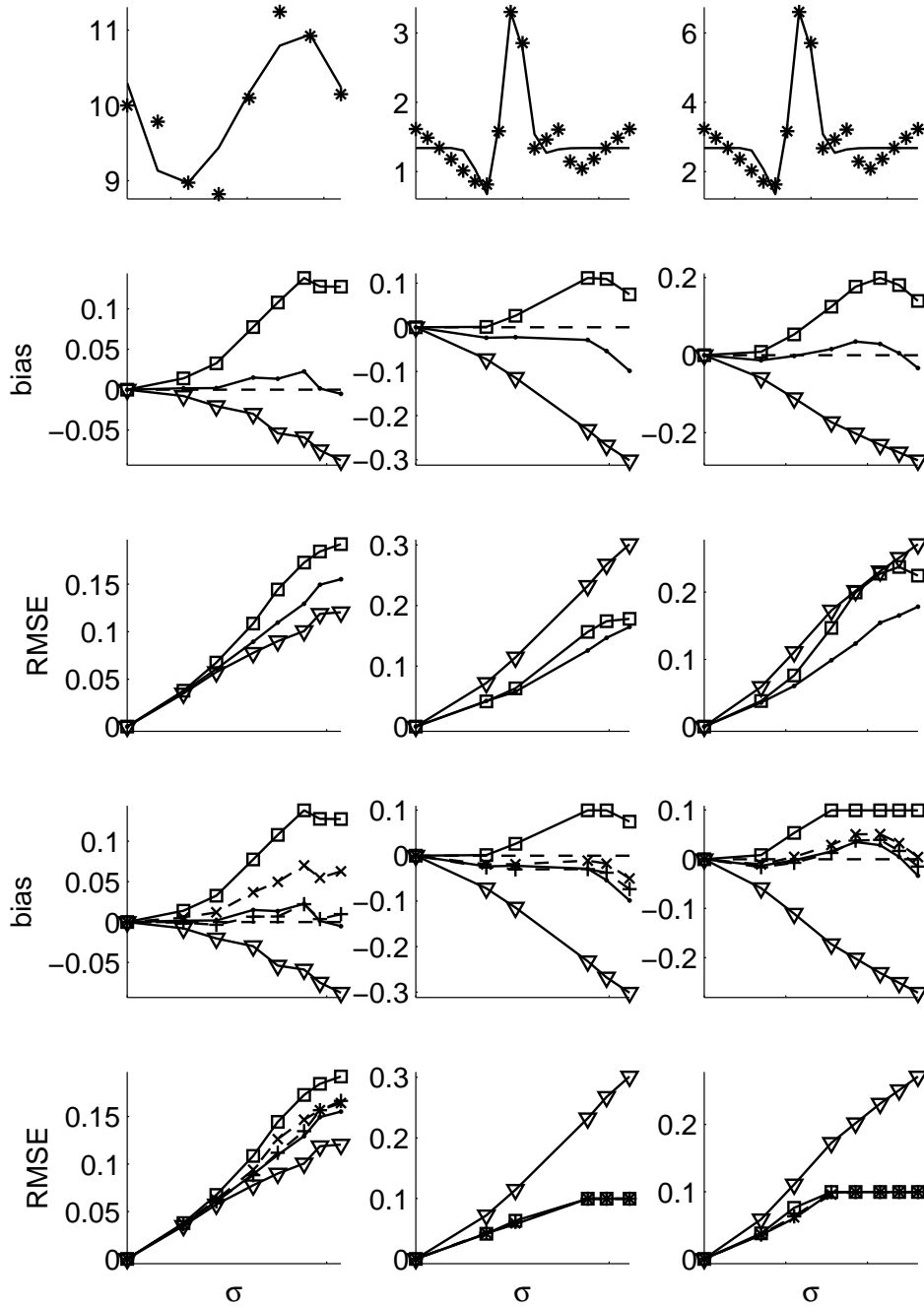

Figure 1: Simulation results: **Left column:** a 3rd degree polynomial is fit to noise data drawn from an underlying sine-function. **Center & Right column:** a Gabor function is fit to noisy data around a linear combination of three Gaussians – two 'excitatory' and one 'inhibitory'. **Left & Center:** Gaussian noise, **Right:** Gamma distributed noise (Fano factor of 2). **First row:** data (stars) and model (lines) are shown in the noise-free condition. Their true VE is 77% and 90%, respectively. **Rows 2-5:** bias (defined as estimated minus true VE) and RMSE are shown as a function of noise $\sigma$. The traditional estimator is shown by triangles, the Sahani-Linden correction by squares, our estimator from eq.(8) by dots. **Rows 4 & 5:** We enforce our prior knowledge that $0 \leq \nu \leq 1$. Estimators with conditioning term $C$ (eq.9) optimized for bias (+) and MSE (x), both dashed, are shown. Restricting VE to $0 \leq \nu \leq 1$ is the reason for the plateau in the bias of the Sahani-Linden estimator (right column, fourth from the top). In all panels data samples with insignificant variation in the data ($p_{\mathrm{ANOVA}} > 0.05$) were excluded from the analysis. Note the different scales in each panel.

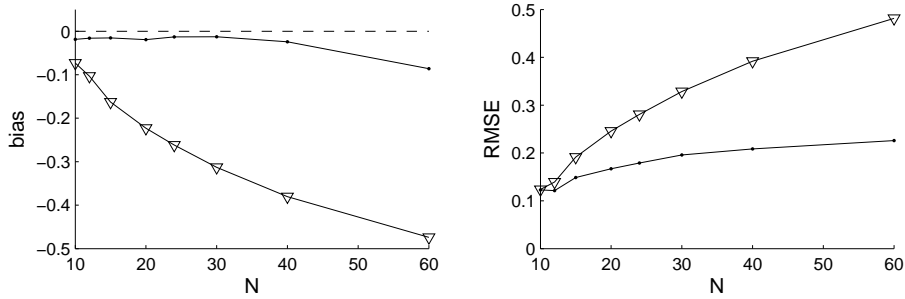

Figure 2: Tradeoff between number of conditions $N$ and number of repetitions $R$ at each condition. Traditional measure: triangles; unbiased estimate: dots. The total number of measurements was fixed at $N \cdot R = 120$, while the number of different conditions $N$ is varied along the abscissa.

at the highest noise levels (at which a large number of data samples does not pass the ANOVA-test for significant modulation), while still remaining smaller than that of the traditional estimator. The reason for the decreasing bias of the Sahani-Linden estimator at very high noise levels is the coincidental cancellation of two bias terms: the negative bias at high noise levels also seen in our estimator for Gabor-fits to differences of Gaussians, and their general positive bias due to not taking the over-fitting of parameters into account. Comparing the MSE (shown as root-mean-square-error or RMSE) of the different estimators shows that they are similar in the case of fitting a polynomial (left column) and significantly improved in the case of fitting a Gabor function (center & right column – note the different $y$-axis scales among all column). [1]

The bottom two rows simulate the situation where our prior knowledge that $0 \leq \text{VE} \leq 1$ is explicitly enforced. Since the numerator in our unbiased estimator (eq.8) yields values around its noiseless value that can be positive and negative, the estimator can be negative or greater than one. Restricting our estimator to $[0..1]$ interferes with its unbiasedness. We test whether a conditioning term can improve the performance of our estimator and find that this is the case for the Gabor fit, but not the polynomial fit. In the case of the Gabor fit, the improvement due to the conditioning term is greatest at the highest noise levels as expected. The bias is decreased at the highest three noise levels tested and the MSE is slightly decreased (at the highest noise level) or the same as with conditioning.

Where the purely analytical formula outperforms the one with conditioning that is because the approximations we have to make in determining the optimal $C$ are greater than the inaccuracy in the analytical formula at those noise levels. This is especially true in the 3rd column where the strongly non-Gaussian noise is incompatible with the Gaussian assumption in our computation of $C$. We conclude that unless one has to estimate VE in the presence of extremely high noise, and has confirmed that conditioning provides an improvement for the particular situation under consideration, our analytical estimator is preferable. (Note the different $y$-axis scales across the 2nd and 4th rows.)

Using an estimator that accounts for the amount of noise has another major benefit. Because the total number of measurements $N \cdot R$ one can make is usually limited, there is a tradeoff between number of conditions $N$ and number of repeats $R$. Everything else being equal the result from the traditional estimator for VE will depend strongly on that choice: the more conditions and the fewer repeats, the higher the standard error of the means $\sigma$ (noise) and hence the lower the estimated VE will be – regardless of the model. Figure 2 demonstrates this behavior in the case of fitting a Gabor to a difference-of-Gaussians exactly as in Figure 1. Keeping the total number of measurements constant, the traditional VE (triangles) decreases drastically as the number of conditions $N$ is increased. The new unbiased estimator (dots) in comparison has a much reduced bias and depends only weakly on $R$. This means that relatively few repeats (but at least 2) are necessary, allowing many more conditions to be tested than previously, hence increasing resolution.

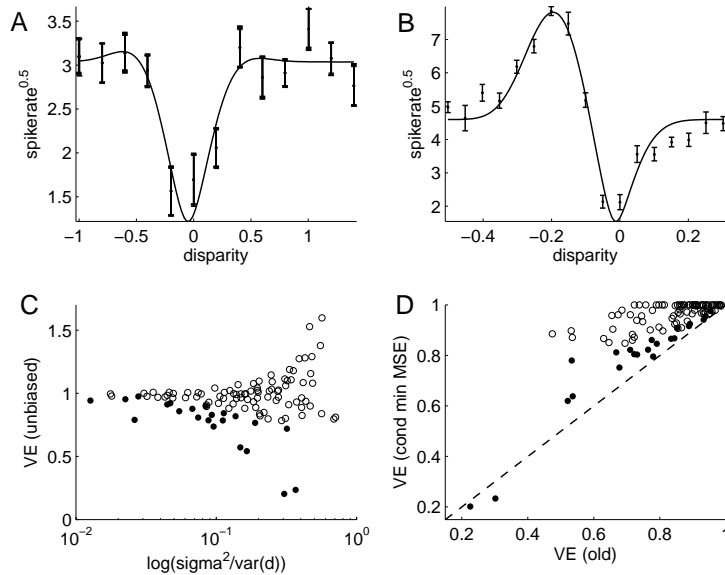

Figure 3: Disparity tuning curves of V1 neurons fit with a Gabor function: **A:** Data from an example neuron shown by their standard error of the mean (SEM) errorbars. Estimate of VE by Gabor fit (solid line) changes from 85% to 93% when noise is adjusted for. **B:** Data from 2nd example neuron. VE of Gabor fit changes from 94% to 95%. $\chi^2-$test on compatibility of data with model: $p_{\chi^2} = 4 \cdot 10^{-4}$. **C:** Unbiased VE as a function of signal-to-noise power. One outlier at (0.93;4.0) not shown. **D:** Traditional VE estimate vs unbiased VE with conditioning to minimize MSE. VE values are limited to 0..1 range. **C & D:** Filled symbols denote cells whose responses are incompatible with the Gabor model, as evaluated by a $\chi^2-$test ($p_{\chi^2} < 0.05$).

# 3 Application to experimental data

## 3.1 Methods

The data are recorded extracellularly from isolated V1 neurons in two awake, fixating rhesus macaque monkeys and have been previously published in [7]. The stimulus consisted of dynamic random dots (RDS) with a binocular disparity applied perpendicular to the preferred orientation of the cell. We only included neurons in the analysis which were significantly modulated by binocular disparity as evaluated by a one-way ANOVA test. 109 neurons passed the test with $p_{\text{ANOVA}} < 0.05$. Since neuronal spike counts are approximately Poisson distributed we perform all subsequent analysis using the square root of the spike rates to approximately equalize variances. We fit a Gabor function with six parameters to the spike rates of each cell and perform a $\chi^2-$ test on the residuals. The minimum number of different conditions $N_{\min} = 13$ and the median number of repeats $\text{median}(R) = 15$.

## 3.2 Results

Most disparity tuning curves in V1 are reasonably well-described by Gabor functions, which explain more than 90% of the variance in two thirds of the neurons [8]. Whether the remaining third reflect a failure of the model or are merely a consequence of noise in the data has been an open question.

Panels A & B in Figure 3 show the responses of two example cells together with their best-fitting Gabor functions. The traditional VE in panel A is only 82% even though the data is not significantly different from the model ($p_{\chi^2} = 0.64$). After adjusting for noise, the unbiased VE becomes 92%, i.e. more than half of the unexplained variance can be attributed to the response variability for each measurement. Panel B shows the opposite situation: 94% of the variance is explained according to the traditional measure and only an additional 1% can be attributed to noise. However, despite

this high VE, since the measurement error is relatively small, the model is rejected with a high significance ($p_{\chi^2} = 4 \cdot 10^{-4}$).

Panel C shows the unbiased estimate of the VE for the entire population of neurons depending on their noise power relative to signal power. At high relative noise levels there is a wide spread of values and for decreasing noise, the VE values asymptote near 1. In fact, the overall population mean for the unbiased VE is 98%, compared with the traditional estimate of 82%. This means that for the entire population, most of the variance previously deemed unexplained by the model can in fact be accounted for by our uncertainty about the data. 22 out of 109 cells or 20% rejected the model ($p_{\chi^2} < 0.05$) and are denoted by filled circles. Panel D demonstrates the effect of the new measure on each individual cell. For the estimation of the true VE for each neuron individually, we incorporate our knowledge about the bounds $0 \leq \nu_0 \leq 1$ and optimize the conditioning term for minimum MSE. With the exception of two neurons, the new estimate of the true VE is greater than the traditional one. On average 40% of the unexplained variance in each individual neuron can be accounted for by noise.

## 4 Conclusions

We have derived an new estimator of the variance explained by models describing noisy data. This estimator improves on previous work in three ways: 1) by accounting for overfitting due to free model parameters, 2) by adjusting for the uncertainty in our estimate of the noise and 3) by describing a way to add an appropriate level of conditioning in cases of very low signal-to-noise in the data or other imposed constraints. Furthermore, our estimator does not rely on a large number of repetitions of the same stimulus in order to perform an extrapolation to zero noise. In numerical simulations with Gaussian and strongly skewed noise we have confirmed that our correction is capable of accounting for most noise levels and provides an estimate with greatly improved bias compared to previous estimators. We note that where the results from the two simulations differ, it is the more realistic simulation where the new estimator performs best.

Another important benefit of our new estimator is that it addresses the classical experimenter's dilemma of a tradeoff between number of conditions $N$ and number of repeats $R$ at each condition. While the results from the traditional estimator quickly deteriorate with increasing $N$ and decreasing $R$, the new estimator is much closer to invariant with respect to both – allowing the experimenter to choose a greater $N$ for higher resolution.

When applying the new VE estimator to a data set of macaque V1 disparity tuning curves we find that almost all of the variance previously unaccounted for by Gabor fits can be attributed to sampling noise. For our population of 109 neurons we find that 98% of the variance can be explained by a Gabor model. This is much higher than previous estimates precisely because they did not account for the variability in their data, illustrating the importance of this correction especially in cases where the model is good. The improvement we present is not limited to neuronal tuning curves but will be valuable to any model testing where noise is an important factor.

**Acknowledgments**

We thank Christian Quaia and Stephen David for helpful discussions.

**References**

[1] S.V. David, and J.L. Gallant, Network 16, 239 (2005).

[2] M. Sahani, and J.F. Linden, Advances in Neural Information Processing Systems 15, 109 (2003).

[3] A. Hsu, A. Borst, and F.E. Theunissen, Network 15, 91 (2004).

[4] C.K. Machens, M.S. Wehr, and A.M. Zador, J Neurosci 24, 1089 (2004).

[5] I. Nauhaus, A. Benucci, M. Carandini, and D.L. Ringach, Neuron 57, 673 (2008).

[6] V. Mante, V. Bonin, and M. Carandini, Neuron 58, 625 (2008).

[7] R.M. Haefner and B.G. Cumming, Neuron 57, 147 (2008).

[8] S.J. Prince, A.D. Pointon, B.G. Cumming, and A.J. Parker, J Neurophysiol 87, 191 (2002).

## Footnotes

[1]It is not surprising that the precise behavior of the respective estimators varies between examples. Two approximations were made in the analytical derivation: (1) the model is approx. linear in its parameters and (2) unbiasing the denominator is not the same as unbiasing the ratio. Both approximations are accurate in the small noise regime. However, as noise levels increase they introduce biases that interact depending on the situation.
